# Max-Margin Structured Output Regression for Spatio-Temporal Action Localization

**Du Tran and Junsong Yuan**
School of Electrical and Electronic Engineering
Nanyang Technological University, Singapore
`trandu@gmail.com, jsyuan@ntu.edu.sg`

## Abstract

Structured output learning has been successfully applied to object localization, where the mapping between an image and an object bounding box can be well captured. Its extension to action localization in videos, however, is much more challenging, because we need to predict the locations of the action patterns both spatially and temporally, i.e., identifying a sequence of bounding boxes that track the action in video. The problem becomes intractable due to the exponentially large size of the structured video space where actions could occur. We propose a novel structured learning approach for spatio-temporal action localization. The mapping between a video and a spatio-temporal action trajectory is learned. The intractable inference and learning problems are addressed by leveraging an efficient Max-Path search method, thus making it feasible to optimize the model over the whole structured space. Experiments on two challenging benchmark datasets show that our proposed method outperforms the state-of-the-art methods.

## 1 Introduction

Blaschko and Lampert have recently shown that object localization can be approached as structured regression problem [2]. Instead of modeling object localization as a binary classification and treating every bounding box independently, their method trains a discriminant function directly for predicting the bounding boxes of objects located in images. Compared with conventional sliding-window based approach, it considers the correlations among the output variables and avoids an exhaustive search of the subwindows for object detection.

Motivated by the successful application of structured regression in object localization [2], it is natural to ask if we can perform structured regression for action localization in videos. Although this idea looks plausible, the extension from object localization to action localization is non-trivial. Different from object localization, where a visual object can be well localized by a 2-dimensional (2D) subwindow, human actions cannot be tightly bounded in such a similar way, i.e., using a 3-dimensional (3D) subvolume. Although many current methods for action detection are based on this 3D subvolume assumption [6, 9, 20, 29], and search for video subvolumes to detect actions, such an assumption is only reasonable for "static" actions, where the subjects do not move globally *e.g.*, pick-up or kiss. For "dynamic" actions, where the subjects can move globally *e.g.*, walk, run, or dive, the subvolume constraint is no longer suitable. Thus, a more accurate localization scheme that can track the actions is required for localizing dynamic actions in videos. For example, one can localize an action by a 2D bounding box in each frame, and track it as the action moves across different frames. This localization structured output generates a smooth spatio-temporal path of connected 2-D bounding boxes. Such a spatio-temporal path can tightly bound the actions in the video space and provides a more accurate spatio-temporal localization of actions.

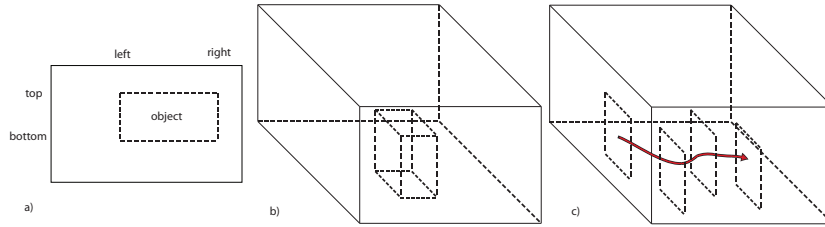

Figure 1: **Complexities of object and action localization**: a) Object localization is of O($n^4$). b) Action localization by subvolume search is of O($n^6$). c) Spatio-temporal action localization in a much larger search space.

However, as the video space is much larger than the image space, spatio-temporal action localization has a much larger structured space compared with object localization. For a video with size $w \times h \times n$, the search space for 3D subvolumes and 2D subwindows is only $O(w^2h^2n^2)$ and $O(w^2h^2)$, respectively (Figure 1). However, the search space for possible spatio-temporal paths in the video space is exponential $O(whnk^n)$[23] if we do not know the start and end points of the path ($k$ is the number of incoming edges per node). Any one of these paths can be the candidates for spatio-temporal action localization, thus an exhaustive search is infeasible. This huge structured space keeps structured learning approaches from being practical to spatio-temporal action localization due to intractable inferences.

This paper proposes a new approach for spatio-temporal action localization which mainly addresses the above mentioned problems. Instead of using the 3D subvolume localization scheme, we precisely locate and track the action by finding an optimal spatio-temporal path to detect and localize actions. The mapping between a video and a spatio-temporal action trajectory is learned. By leveraging an efficient Max-Path search method [23], the intractable inference and learning problems can be addressed, thus makes our approach practical and effective although the structured space is very large. Being solved as structured learning problem, our method can well exploit the correlations between local dependent video features, and therefore optimizes the structured output. Experiments on two challenging benchmark datasets show that our method significantly outperforms the state-of-the-art methods.

## 1.1 Related work

Human action detection is traditionally approached by spatio-temporal video volume matching using different features: space-time orientation [6], volumetric [9], action MACH [20], HOG3D [10]. The sliding window scheme is then applied to locate actions which is ineffective and time-consuming. Different matching, learning models have also been introduced. Boiman and Irani proposed ensembles of patches to detect irregularities in images and videos [3]. Hu *et al* used multiple-instance learning to detect actions [8]. Mahadevan *et al* used mixtures of dynamic textures to detect anomaly events [15]. Le *et al* used deep learning to learn unsuppervised features for recognizing human activities [14]. Niebles *et al* used a probabilistic latent semantic analysis model for recognizing actions [17]. Yao et al trained probabilistic non-linear latent variable models to track complex activities [28]. Yuan *et al* extended the branch-and-bound subwindow search [11] to subvolume search for action detection [29]. Recently, Tran and Yuan relaxed the 3D bounding box constraint for detecting and localizing medium and long video events [23]. Despite the improvements over 3D subvolume based approaches, this method did not fully utilize the correlations between local part detectors as they were independently trained.

Max-margin structured output learning [19, 21, 24] was recently proposed and demonstrated its success in many applications. One of its attractive features is that although the structured space can be very large, whenever inference is tractable, learning is also tractable. Finley and Joachims further showed that overgenerating (*e.g.* relaxations) algorithms have theoretic advantages over undergenerating (*e.g.* greedy) methods when exact inference is intractable [7]. Various structured learning based approaches were proposed to solve computer vision problems including pedestrian detection [22], object detection [2, 25], object segmentation [1], facial action unit detection [16], human interaction recognition [18], group activity recognition [13], and human pose parsing [27]. More recently, Lan *et al* used a latent SVM to jointly detect and recognize actions in videos [12].

Among these work, Lan *et al* is most similar to ours. However, this method requires a reliable human detector in both inference and learning, thus it is not applicable to "dynamic" actions where the human poses are significantly varied. Moreover, because of its using HOG3D [26], it only detects actions in a sparse subset of frames where the interest points present.

## 2 Spatio-Temporal Action Localization as Structured Output Regression

Given a video $x$ with the size of $w \times h \times m$ where $w \times h$ is the frame size and $m$ is its length, to localize actions, one needs to predict a structured object $y$ which is a smooth spatio-temporal path in the video space. We denote a path $y = \{(l, t, r, b)_{i=1..m}\}$ where $(l, t, r, b)_i$ are respectively the *left, top, right, bottom* of the rectangle that bounds the action in the $i$-th frame. These values of $(l, t, r, b)$ are all set to zeros when there is no action in this frame. Because of the spatio-temporal smoothness constraint, the boxes in $y$ are necessarily smoothed over the spatio-temporal video space. Let us denote $\mathcal{X} \subset [0, 255]^{3whm}$ as the set of color videos, and $\mathcal{Y} \subset \mathbb{R}^{4m}$ as the set of all smooth spatio-temporal paths in the video space. The problem of *spatio-temporal action localization* becomes to learn a structured prediction function of $f : \mathcal{X} \mapsto \mathcal{Y}$.

### 2.1 Structured Output Learning

Let $\{x_1, \ldots, x_n\} \subset \mathcal{X}$ be the training videos, and $\{y_1, \ldots, y_n\} \subset \mathcal{Y}$ be their corresponding annotated ground truths. We formulate the action localization problem using the structured learning as presented in [24]. Instead of searching for $f$, we learn a *discriminant function* $F : \mathcal{X} \times \mathcal{Y} \mapsto \mathbb{R}$. $F$ is a compatibility function which measures how compatible the localization $y$ will be suited to the given input video $x$. If the model utilizes a parameter set of $w$, then we denote $F(x, y; w) = \langle w, \phi(x, y) \rangle$, which is a family of functions parameterized by $w$, and $\phi(x, y)$ is a joint kernel feature map which represents spatio-temporal features of $y$ given $x$.

Once $F$ is trained, meaning the optimal parameter $w^*$ is determined, the final prediction $\hat{y}$ can be obtained by maximizing $F$ over $\mathcal{Y}$ for a specific input $x$.

$$\hat{y} = f(x; w^*) = \underset{y \in \mathcal{Y}}{\operatorname{argmax}} F(x, y; w^*) = \underset{y \in \mathcal{Y}}{\operatorname{argmax}} \langle w^*, \phi(x, y) \rangle \tag{1}$$

The optimal parameter set $w^*$ is selected by solving the convex optimization problem in Eq. 2:

$$\begin{aligned}
\min_{w, \xi} \quad & \frac{1}{2}\|w\|^2 + C \sum_{i=1}^{n} \xi_i \\
\text{s.t.} \quad & \langle w, \phi(x_i, y_i) - \phi(x_i, y) \rangle \geq \Delta(y_i, y) - \xi_i, \forall i, \forall y \in \mathcal{Y} \backslash y_i, \\
& \xi_i \geq 0, \forall i.
\end{aligned} \tag{2}$$

Eq. 2 optimizes $w$ such that the score of the true structure $y_i$ of $x_i$ will be larger than any other structure $y$ by a margin which is rescaled by the loss of $\Delta(y_i, y)$. The loss function will be defined in Section 2.3. This optimization is similar to the traditional support vector machine (SVM) formulation except for two differences. First, the number of constraints is much larger due to the huge size of the structure space $\mathcal{Y}$. Second, the margins are rescaled differently by the constraint's loss $\Delta(y_i, y)$. Because of the large number of constraints, the problem in Eq. 2 cannot be solved directly although it is a convex problem. Alternatively, one can solve the above problem by the cutting plane algorithm [24] or subgradient methods [19, 21]. We use the cutting plane algorithm to solve this learning problem. The algorithm starts with a random parameter $w$ and an empty constraint set. At each round, it searches for the most violated constraint and add it to the constraint set. This step is to search for $y$ that maximizes the violation value $\xi_i$ (Eq. 3). When a new constraint is found, the optimization is applied to update $w$. The process is repeated until no more constraint is added. This algorithm is proven to converge [24] and normally within a small number of constraints due to the sparsity of the structured space.

$$\xi_i \geq \Delta(y_i, y) + \langle w, \phi(x_i, y) \rangle - \langle w, \phi(x_i, y_i) \rangle, \forall y \in \mathcal{Y} \backslash y_i \tag{3}$$

### 2.2 The Joint Kernel Feature Map for Action Localization

Let us denote $x|_y$ as the video portion cut out from $x$ by the path $y$, namely the stack of images cropped by the bounding boxes $b_{1..m}$ of $y$. We also denote $\varphi(b_i) \in \mathbb{R}^k$ as a feature map for a 2D

box $b_i$. It is worth noting that $\varphi(b_i)$ can be represented by either local features (*e.g.* local interest points) or global features (*e.g.* HOG, HOF) of the whole box $b_i$. We thus have a feature map for $x|_y$ as $\phi(x, y)$ which is also a vector in $\mathbb{R}^k$:

$$\phi(x, y) = \frac{1}{m} \sum_{i=1}^{m} \varphi(b_i) \tag{4}$$

Finally, the decision function of our structured prediction is now formed as in Eq. 5.

$$F(x, y; w) = \langle w, \phi(x, y) \rangle = \frac{1}{m} \sum_{i=1}^{m} \langle w, \varphi(b_i) \rangle. \tag{5}$$

## 2.3 Loss Function

We define a Hinge loss function $\Delta : \mathcal{Y} \times \mathcal{Y} \mapsto [0, 1]$ for evaluating the loss induced by a predicted structure $\hat{y}$ compared with a true structure label $y$. We denote $y = \{b_{i=1..m}\}$, where $b_i = (l, t, r, b)_i$ is the ground truth box of the $i$-th frame. Similarly, we denote $\hat{y} = \{\hat{b}_{i=1..m}\}$ the predicted structure. The loss function is defined as follow:

$$\Delta(y, \hat{y}) = \frac{1}{m} \sum_{i=1}^{m} \delta(b_i, \hat{b}_i). \tag{6}$$

$$\delta(b, \hat{b}) = \begin{cases} 1 - \frac{Area(b \cap \hat{b})}{Area(b \cup \hat{b})} & \text{if } l_b = l_{\hat{b}} = 1 \\ 1 - (\frac{1}{2}(l_b l_{\hat{b}} + 1)), & \text{otherwise.} \end{cases} \tag{7}$$

$$l_b = \begin{cases} -1 & \text{if } b = (0, 0, 0, 0) \\ 1, & \text{otherwise.} \end{cases} \tag{8}$$

## 3 Inference and Learning

We need a feasible way to perform the inference in Eq. 1 during testing which can be rewritten as in Eq. 9.

$$\hat{y} = \underset{y \in \mathcal{Y}}{\operatorname{argmax}} \langle w, \phi(x, y) \rangle = \frac{1}{m} \underset{y \in \mathcal{Y}}{\operatorname{argmax}} \sum_{i=1}^{m} \langle w, \varphi(b_i) \rangle. \tag{9}$$

During training, we need to search for the most violated constraints by maximizing the right hand side of Eq. 3 which is equivalent to Eq. 10. From now on, we denote $\bar{y}$ for $y_i$ in Eq. 2 because the example index $i$ is no longer important.

$$\max_{y \in \mathcal{Y}} \{ \Delta(y, \bar{y}) + \langle w, \phi(x, y) \rangle \} \tag{10}$$

$$= \max_{y \in \mathcal{Y}} \left\{ \frac{1}{m} \sum_{i=1}^{m} \delta(b_i, \bar{b}_i) + \frac{1}{m} \sum_{i=1}^{m} \langle w, \varphi(b_i) \rangle \right\} \tag{11}$$

$$= \frac{1}{m} \max_{y \in \mathcal{Y}} \left\{ \sum_{i=1}^{m} \left( \delta(b_i, \bar{b}_i) + \langle w, \varphi(b_i) \rangle \right) \right\} \tag{12}$$

To solve Eq. 9 and Eq. 12, one needs to search for a smooth path $y^*$ in the spatio-temporal video space $\mathcal{Y}$ which gives the maximum total score. Both of the above equations are difficult due to the large size of $\mathcal{Y}$, *e.g.* the exponential number of possible spatio-temporal paths in $\mathcal{Y}$ (see supplemental material). We now show that both problems in Eq. 9 and Eq. 12 can be reduced to Max-Path search problem and solved by [23] efficiently. Max-Path algorithm [23] was proposed to detect dynamic video events. It is guaranteed to obtain the best spatio-temporal path in the video space provided that the local windows' scores can be precomputed. The algorithm takes a 3D trellis of local windows' scores as input, and outputs the best path which the maximum total score. In testing, the trellis's local scores are $\langle w, \varphi(b_i) \rangle$ where $b_i$ is the local window. These values are easily evaluated given a $w$ and a feature map $\varphi$. In training, those values of the trellis are $\delta(b_i, \bar{b}_i) + \langle w, \varphi(b_i) \rangle$ which are also computable given parameter $w$, feature map $\varphi$, and ground truth $\bar{b}_i$. After the trellis is constructed, the Max-Path algorithm is employed to find the best path, therefore we can identify the smoothed spatio-temporal path $y^*$ that maximizes Eq. 9 and Eq. 12.

### 3.1 Constraint Enforcement

Let us consider one constraint in Eq. 2, here we ignore the index $i$ of the example for simplicity and use $\bar{y}$ as the ground truth for example $x$. We also denote $y = b_{1..m}$ and $\bar{y} = \bar{b}_{1..m}$.

$$\langle w, \phi(x, \bar{y}) \rangle - \langle w, \phi(x, y) \rangle \geq \Delta(\bar{y}, y) - \xi, \forall y \in \mathcal{Y} \backslash \bar{y} \tag{13}$$

$$\Leftrightarrow \quad \frac{1}{m} \sum_{i=1}^{m} \langle w, \varphi(\bar{b}_i) \rangle - \frac{1}{m} \sum_{i=1}^{m} \langle w, \varphi(b_i) \rangle \geq \frac{1}{m} \sum_{i=1}^{m} \delta(b_i, \bar{b}_i) - \xi, \forall y \in \mathcal{Y} \backslash \bar{y} \tag{14}$$

$$\Leftrightarrow \quad \sum_{i=1}^{m} \langle w, \varphi(\bar{b}_i) \rangle - \sum_{i=1}^{m} \langle w, \varphi(b_i) \rangle \geq \sum_{i=1}^{m} \delta(b_i, \bar{b}_i) - m\xi, \forall y \in \mathcal{Y} \backslash \bar{y} \tag{15}$$

The constraint in Eq. 15 can be split into $m$ constraints in Eq. 16 which are harder, therefore satisfying these $m$ constraints will lead to satisfying the Eq. 15 constraint

$$\langle w, \varphi(\bar{b}_i) \rangle - \langle w, \varphi(b_i) \rangle \geq \delta(b_i, \bar{b}_i) - \xi, \forall i \in [1..m], \forall y \in \mathcal{Y} \backslash \bar{y} \tag{16}$$

In training, instead of solving Eq. 2 with the constraints in Eq. 13, we solve it with the set of constraints as in Eq. 16. The problem is harder because of tighter constraints. However, the important benefit of using such enforcements is that instead of comparing features of two different spatio-temporal paths $y$ and $\bar{y}$, one can compare the features of individual box pairs $(b_i, \bar{b}_i)$ of those two paths. This constraint enforcement will help the training algorithm to avoid comparing features of two paths of different lengths which is unstable due to feature normalization.

## 4 Experimetial Setup

**Datasets**: we conduct experiments on two datasets: *UCF-Sport* [20] and *Oxford-TV* [18]. UCF-Sport dataset consists of 150 video sequences of 10 different action classes. We use the same split as in [12] for training and testing. On this dataset, we detect three different actions: horse-riding, running, and diving. We choose those actions because they have different levels of body movements. Horse riding is relatively rigid; running is more deformable; while diving is extremely deforming in terms of articulated body movements. Oxford-TV dataset consists of 300 videos taken from real TV programs. It has 4 classes of actions: hand-shake, high-five, hug, kiss, and a set of 100 negative videos. As used in [18], this dataset is divided into two equal subsets. We use set 1 for training and set 2 for testing. We perform the task of kiss detection and localization on this dataset. Kissing actions is more challenging compared with other action classes in this dataset due to less motion and appearance cues.

**Features and Parameters**: our algorithm needs a feature representation $\varphi(b)$ of a cropped image $b$. We use a global representation for $\varphi(b)$ using Histogram of Oriented Gradients (HOF) [4] and Histogram of Flows (HOF) [5]. The cropped image $b$ is divided into $h \times v$ half-overlapped blocks; each block has $2 \times 2$ cells. Each cell is represented by a 9-bin histogram. The feature vector's length become $h \times v \times 2 \times 2 \times 9 \times 2 = 72 \times h \times v$ for both HOG and HOF. $(h, v)$ can be different for each class due to different shape-ratios of the actions (*e.g.* rectangle boxes for horse-riding and running, square boxes for diving). More specifically, we use $(7, 15)$ for horse-riding and running, $(11, 11)$ for diving, $(9, 7)$ for kissing. The regularization parameter $C$ in Eq. 2 is set to 1 for all cases.

**Evaluation Metrics**: we quantitatively evaluate different methods in both detection and localization. As used in [12], the video localization score is measured by averaging its frame localization scores which are the overlap area divided by the union area of the predicted and truth boxes. A prediction is then considered as correct if its localization score is greater or equal to $\sigma = 0.2$. It is worth noting that detection evaluations are applied to both positive and negative testing examples while localization evaluations are only applied to positive ones. As a result, the detection metric is to measure the reliability of the detections (precision/recall) where the localization metric indicates the quality of detections, *e.g.* how accurate are the predicted spatio-temporal paths compared with ground truth. More specific, detection is to answer the question "Is there any action of interest in this video?" while localization is to answer to "Provided that there is one action instance that appears in this video, where is it?".

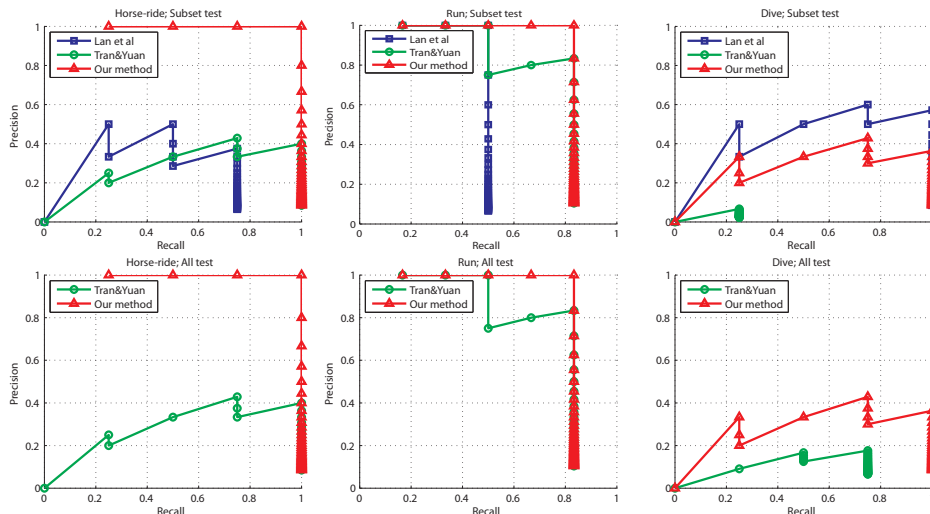

Figure 2: **Action detection results on UCF-Sport**: detection curves of our proposed method compared with [12] and [23]. Upper plots are detection results evaluated on subset frames given by [12], while lower plots are the results of all-frame evaluations. Except for diving, our proposed method significantly improves the other methods.

| Eval. Set | Method | H-Ride | Run | Dive | Average |
|-----------|--------|--------|-------|--------|---------|
| Subset | [12] | 21.75 | 19.60 | **42.67** | 28.01 |
| | [23] | 62.19 | 50.20 | 16.41 | 42.93 |
| | Our | **68.06** | **61.41** | 36.54 | **55.34** |
| All | [12] | N/A | N/A | N/A | N/A |
| | [23] | 63.06 | 48.09 | 22.64 | 44.60 |
| | Our | **64.01** | **61.86** | **37.03** | **54.30** |

Table 1: **Action localization results on UCF-Sport**: comparisons among our proposed method, [12], and [23]. The upper section presents results evaluated on a subset of frames given by [12], while the lower section reports results from evaluating on all frames. Our method improves 27.33% from [12] and 12.41% from [23] on subset evaluations and improves 9.7% from [23] on all-frame evaluations. N/A indicates not applicable.

## 5   Experimental Results

**UCF-Sport**: we compare our method with two current approaches: Lan *et al* [12], Tran and Yuan [23]. The output predictions of Lan *et al* are directly obtained from [12]. For [23], we train a linear SVM detector for each action class using the same features as ours. The Max-Path algorithm is then applied to detect the actions of interest. According to [12], its method used HOG3D [26], so that it is only able to detect and localize actions at a sparse set of frames where the HOG3D interest points present. To provide a fair comparison with [12], we report two different sets of evaluations. The first set is applied only to the subset of frames where [12] reports detections and the second set is to take all frames into consideration.

Table 1 reports the results of action localization of different methods and action classes. On average, our method improves 27.33% from [12] and 12.41% from [23] on subset evaluations and improves 9.7% from [23] on all-frame evaluations. Figure 2 shows detection results of different methods on UCF-Sport dataset. Our method significantly improves over [23] for all three action classes on both subset and all-frame evaluations. Compared with [12] on subset evaluations, our method significantly improve over [12] on horse-riding and running detection. However, [12] provides better detection results than ours on diving detection. This better detection is because their interest-point-based sparse features are more suitable to deformable actions as diving. For a complete presentation, we visualze localization results of our method comapred with those of [12] and [23] on a diving sequence (Figure 3). All predicted boxes are plotted together with ground truth boxes for comparisons. It is worth noting that [12] has only predictions at a sparse set of frames, therefore blue

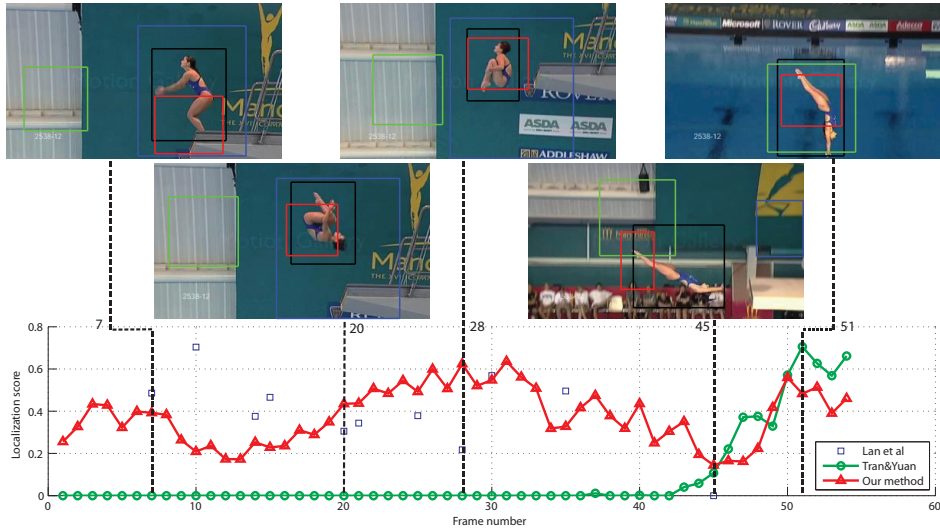

Figure 3: **Visualization of diving localization**: the plots of localization scores of different methods on a diving video sequence. Lan et al's [12] results are visualized in blue, Tran and Yuan's [23] are green, ours are red, and ground truth are black boxes. Best view in color.

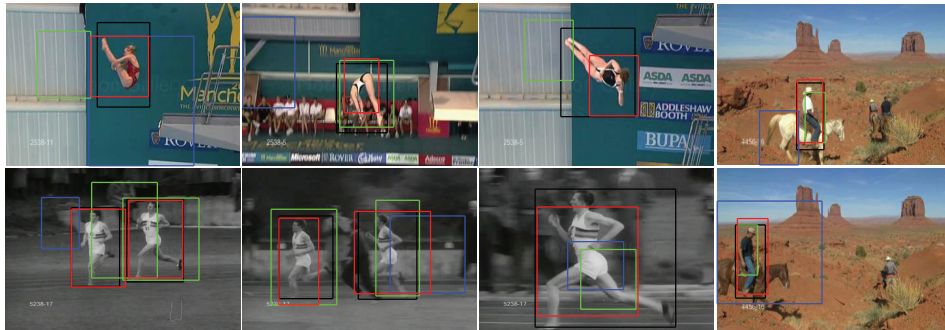

Figure 4: **Action detection and localization on UCF-Sport**: Lan et al's [12] results are visualized in blue, Tran and Yuan's [23] are green, ours are read, and ground truth are black. Our method and [23] can detect multiple instances of actions (two bottom left images).

squares are visualized as discrete dots while the other methods are visualized by continuous curves. Our method (red curve) localizes diving action much more accurately than [23] (green curve). [12] localizes diving action fairly good, however it is not applicable when more accurate localizations (*e.g.* all frame predictions) are required.

**Oxford-TV**: we compare our method with [23] on both detection and localization tasks. For detection, we report two different quantitative evaluations: the equal precision-recall (EPR) and the area under ROC curve (AUC). For localization, besides the spatial localization (SL) metric as used in UCF dataset experiments, we also evaluate different methods by temporal localization (TL) metric. This metric is not applicable to UCF dataset because most action instances in UCF dataset start and end at the first and last frame, respectively. Temporal localization is computed as the length

| Method | EPR(%) | AUC | SL(%) | TL(%) |
|--------|--------|-----|-------|-------|
| [18]   | 32.50* | N/A | N/A   | N/A   |
| [23]   | 24.14  | 0.27 | 18.46 | 40.09 |
| Our    | **38.89** | **0.42** | **39.52** | **45.30** |

Table 2: **Kiss detection and localization results**. We improve 14.74% in equal precision/recall detection rate, 0.15 in area under ROC curve, 21.06% in spatial localization, and 5.21% in temporal localization over [23]. *Result of [18] is not directly comparable. N/A indicates not applicable.

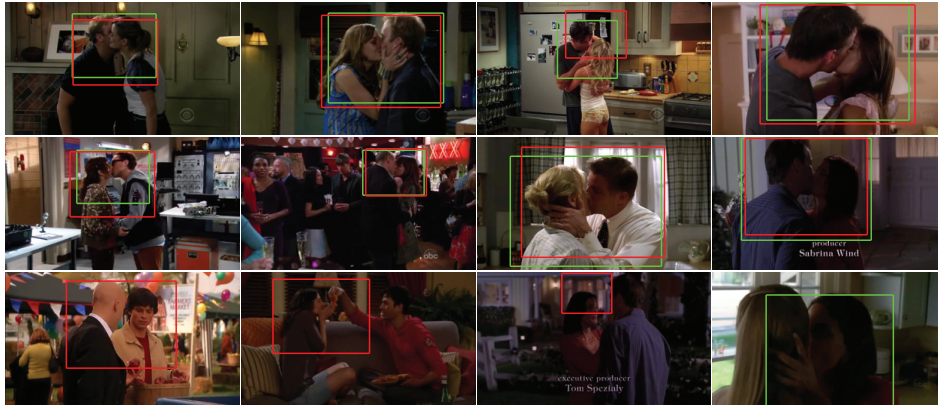

Figure 5: **Visualizaiton of kiss detection**: our results are visualized in red; ground truths are in green. The upper two rows are some of correct detections while the last row shows false or missed detections.

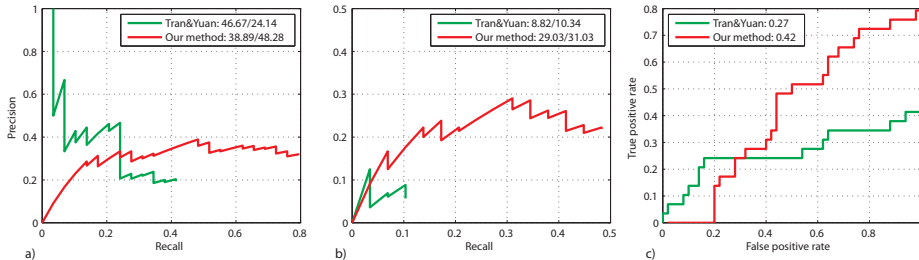

Figure 6: **Kiss detection results**: a) Precision-recall curves with $\sigma = 0.2$. b) Precision-recall curves with $\sigma = 0.4$. c) ROC curves with $\sigma = 0.2$. Numbers inside the legends are best precision-recall values(a and b) and the area under ROC curve(c).

(measured in frames) of the intersection divided by the union of detection and ground truth. Table 2 presents detection and localization results of our proposed method compared with [23]. On localization task, our method improves 21.06% in spatial localization, and 5.21% in temporal localization over [23]. On detection task, by using the cut-off threshold $\sigma = 0.2$, our method improves 14.74% in equal precision-recall rate and 0.15 in area under ROC curve over [23] (Figure 6a and 6c). One may further ask "what if we need more accurate detections?". Interestingly, when we increase the cut-off threshold $\sigma$ to 0.4, [23] significantly drops from 24.11% to 8.82% while our method remains 29.03% (Figure 6b) which demonstrates that our method can simultaneously detect and localize actions with high accuracy.

## 6 Conclusions

We have proposed a novel structured learning approach for spatio-temporal action localization in videos. While most of current approaches detect actions as 3D subvolumes [6, 9, 20, 29] or a sparse subset of frames [12], our method can precisely detect and track actions in both spatial and temporal spaces. Although [23] is also applicable to spatio-temporal action detection, this method cannot be optimized over the large video space due to its independently trained detectors. Our approach significantly outperforms [23] thanks to the structured optimization. This improvement gap is also consistent with the theoretic analysis in [7]. Moreover, being free from people detection and background subtraction, our approach can efficiently handle unconstrained videos and be easily extended to detect other spatio-temporal video patterns. Strong experimental results on two challenging benchmark datasets demonstrate that our proposed method significantly outperforms the state-of-the-arts.

**Acknowledgments**

The authors would like to thank Tian Lan for reproducing [12]'s results on UCF dataset, Minh Hoai Nguyen for useful discussions about the cutting-plane algorithm. This work is supported in part by the Nanyang Assistant Professorship (SUG M58040015) to Dr. Junsong Yuan.

# References

[1] L. Bertelli, T. Yu, D. Vu, and S. Gokturk. Kernelized structural SVM learning for supervised object segmentation. *CVPR*, 2011.

[2] M. B. Blaschko and C. H. Lampert. Learning to localize objects with structured output regression. *ECCV*, 2008.

[3] O. Boiman and M. Irani. Detecting irregularities in images and in video. *IJCV*, 2007.

[4] N. Dalal and B. Triggs. Histograms of oriented gradients for human detection. *CVPR*, 2005.

[5] N. Dalal, B. Triggs, and C. Schmid. Human detection using oriented histograms of flow and appearance. *ECCV*, 2006.

[6] K. Derpanis, M. Sizintsev, K. Cannons, and P. Wildes. Efficient action spotting based on a spacetime oriented structure representation. *CVPR*, 2010.

[7] T. Finley and T. Joachims. Training structural SVMs when exact inference is intractable. *ICML*, 2008.

[8] Y. Hu, L. Cao, F. Lv, S. Yan, Y. Gong, and T. S. Huang. Action detection in complex scenes with spatial and temporal ambiguities. *ICCV*, 2009.

[9] Y. Ke, R. Sukthankar, and M. Hebert. Volumetric features for video event detection. *IJCV*, 2010.

[10] A. Klaser, M. Marszalek, and C. Schmid. A spatio-temporal descriptor based on 3d-gradients. *BMVC*, 2008.

[11] C. H. Lampert, M. B. Blaschko, and T. Hofmann. Efficient subwindow search: A branch and bound framework for object localization. *IEEE Trans. on Pattern Analysis and Machine Intelligence*, 2009.

[12] T. Lan, Y. Wang, and G. Mori. Discriminative figure-centric models for joint action localization and recognition. *ICCV*, 2011.

[13] T. Lan, Y. Wang, W. Yang, and G. Mori. Beyond actions: Discriminative models for contextual group activities. *NIPS*, 2010.

[14] Q. Le, W. Zou, S. Yeung, and A. Ng. Learning hierarchical spatio-temporal features for action recognition with independent subspace analysis. *CVPR*, 2011.

[15] V. Mahadevan, W. Li, V. Bhalodia, and N. Vasconcelos. Anomaly detection in crowded scenes. *CVPR*, 2010.

[16] M. H. Nguyen, T. Simon, F. De la Torre, and J. Cohn. Action unit detection with segment-based SVMs. *CVPR*, 2010.

[17] J. C. Niebles, H. Wang, and L. Fei-Fei. Unsupervised learning of human action categories using spatial-temporal words. *International Journal of Computer Vision*, 2008.

[18] A. Patron-Perez, M. Marszalek, A. Zisserman, and I. Reid. High five: Recognising human interactions in tv shows. *BMVC*, 2010.

[19] N. Ratliff, J. A. Bagnell, and M. Zinkevich. Subgradient methods for maximum margin structured learning. *ICML 2006 Workshop on Learning in Structured Output Spaces*, 2006.

[20] M. D. Rodriguez, J. Ahmed, and M. Shah. Action mach: A spatio-temporal maximum average correlation height filter for action recognition. *CVPR*, 2008.

[21] B. Taskar, S. Lacoste-Julien, and M. Jordan. Structured prediction via the extragradient method. *NIPS*, 2005.

[22] D. Tran and D. Forsyth. Configuration estimates improve pedestrian finding. *NIPS*, 2007.

[23] D. Tran and J. Yuan. Optimal spatio-temporal path discovery for video event detection. *CVPR*, pages 3321–3328, 2011.

[24] I. Tsochantaridis, T. Joachims, T. Hofmann, and Y. Altun. Large margin methods for structured and interdependent output variables. *JMLR*, 2005.

[25] A. Vedaldi and A. Zisserman. Structured output regression for detection with partial truncation. *NIPS*, 2009.

[26] H. Wang, M. M. Ullah, A. Klaser, I. Laptev, and C. Schmid. Evaluation of local spatio-temporal features for action recognition. *BMVC*, 2009.

[27] Y. Wang, D. Tran, and Z. Liao. Learning hierarchical poselets for human parsing. *CVPR*, 2011.

[28] A. Yao, J. Gall, L. V. Gool, and R. Urtasun. Learning probabilistic non-linear latent variable models for tracking complex activities. *NIPS*, 2011.

[29] J. Yuan, Z. Liu, and Y. Wu. Discriminative video pattern search for efficient action detection. *IEEE Trans. on Pattern Analysis and Machine Intelligence*, 2011.

